# Distribution-Calibrated Hierarchical Classification

**Ofer Dekel**
Microsoft Research
One Microsoft Way, Redmond, WA 98052, USA
oferd@microsoft.com

## Abstract

While many advances have already been made in hierarchical classification learning, we take a step back and examine how a hierarchical classification problem should be formally defined. We pay particular attention to the fact that many arbitrary decisions go into the design of the label taxonomy that is given with the training data. Moreover, many hand-designed taxonomies are unbalanced and misrepresent the class structure in the underlying data distribution. We attempt to correct these problems by using the data distribution itself to calibrate the hierarchical classification loss function. This distribution-based correction must be done with care, to avoid introducing unmanageable statistical dependencies into the learning problem. This leads us off the beaten path of binomial-type estimation and into the unfamiliar waters of geometric-type estimation. In this paper, we present a new calibrated definition of statistical risk for hierarchical classification, an unbiased estimator for this risk, and a new algorithmic reduction from hierarchical classification to cost-sensitive classification.

## 1 Introduction

Multiclass classification is the task of assigning labels from a predefined label-set to instances in a given domain. For example, consider the task of assigning a topic to each document in a corpus. If a training set of labeled documents is available, then a multiclass classifier can be trained using a supervised machine learning algorithm. Often, large label-sets can be organized in a taxonomy. Examples of popular label taxonomies are the ODP taxonomy of web pages [2], the gene ontology [6], and the LCC ontology of book topics [1]. A taxonomy is a hierarchical structure over labels, where some labels define very general concepts, and other labels define more specific specializations of those general concepts. A taxonomy of document topics could include the labels MUSIC, CLASSICAL MUSIC, and POPULAR MUSIC, where the last two are special cases of the first. Some label taxonomies form trees (each label has a single parent) while others form directed acyclic graphs. When a label taxonomy is given alongside a training set, the multiclass classification problem is often called a *hierarchical classification* problem. The label taxonomy defines a structure over the multiclass problem, and this structure should be used both in the formal definition of the hierarchical classification problem, and in the design of learning algorithms to solve this problem.

Most hierarchical classification learning algorithms treat the taxonomy as an indisputable definitive model of the world, never questioning its accuracy. However, most taxonomies are authored by human editors and subjective matters of style and taste play a major role in their design. Many arbitrary decisions go into the design of a taxonomy, and when multiple editors are involved, these arbitrary decisions are made inconsistently. Figure 1 shows two versions of a simple taxonomy, both equally reasonable; choosing between them is a matter of personal preference. Arbitrary decisions that go into the taxonomy design can have a significant influence on the outcome of the learning algorithm [19]. Ideally, we want learning algorithms that are immune to the arbitrariness in the taxonomy.

The arbitrary factor in popular label taxonomies is a well-known phenomenon. [17] gives the example of the *Library of Congress Classification* system (LCC), a widely adopted and constantly updated taxonomy of "all knowledge", which includes the category WORLD HISTORY and four of its direct subcategories: ASIA, AFRICA, NETHERLANDS, and BALKAN PENINSULA. There is a clear imbalance between the the level of granularity of ASIA versus its sibling BALKAN PENINSULA. The *Dewey Decimal Classification* (DDC), another widely accepted taxonomy of "all knowledge", defines ten main classes, each has exactly ten subclasses, and each of those again has exactly ten subclasses. The rigid choice of a decimal fan-out is an arbitrary one, and stems from an aesthetic ideal rather than a notion of informativeness. Incidentally, the ten subclasses of RELIGION in the DDC include six categories about Christianity and the additional category OTHER RELIGIONS, demonstrating the editor's clear subjective predilection for Christianity. The ODP taxonomy of web-page topics is optimized for navigability rather than informativeness, and is therefore very flat and often unbalanced. As a result, two of the direct children of the label GAMES are VIDEO GAMES (with over $42,000$ websites listed) and PAPER AND PENCIL GAMES (with only $32$ websites). These examples are not intended to show that these useful taxonomies are flawed, they merely demonstrate the arbitrary subjective aspect of their design.

Our goal is to define the problem such that it is invariant to many of these subjective and arbitrary design choices, while still exploiting much of the available information. Some older approaches to hierarchical classification do not use the taxonomy in the definition of the classification problem [12, 13, 18, 9, 16]. Namely, these approaches consider all classification mistakes to be equally bad, and use the taxonomy only to the extent that it reduces computational complexity and the number of classification mistakes. More recent approaches [3, 8, 5, 4] exploit the label taxonomy more thoroughly, by using it to induce a hierarchy-dependent loss function, which captures the intuitive idea that not all classification mistakes are equally bad: incorrectly classifying a document as CLASSICAL MUSIC when its true topic is actually JAZZ is not nearly as bad as classifying that document as COMPUTER HARDWARE. When this interpretation of the taxonomy can be made, ignoring it is effectively wasting a valuable signal in the problem input. For example, [8] define the loss of predicting a label $u$ when the correct label is $y$ as the number of edges along the path between the two labels in the taxonomy graph.

Additionally, a taxonomy provides a very natural framework for balancing the tradeoff between specificity and accuracy in classification. Ideally, we would like our classifier to assign the most specific label possible to an instance, and the loss function should reward it adequately for doing so. However, when a specific label cannot be assigned with sufficiently high confidence, it is often better to fall-back on a more general correct label than it is to assign an incorrect specific label. For example, classifying a document on JAZZ as the broader topic MUSIC is better than classifying it as the more specific yet incorrect topic COUNTRY MUSIC. A hierarchical classification problem should be defined in a way that penalizes both over-confidence and under-confidence in a balanced way.

The graph-distance based loss function introduced by [8] captures both of the ideas mentioned above, but it is very sensitive to arbitrary choices that go into the taxonomy design. Once again consider the example in Fig. 1: each hierarchy would induce a different graph-distance, which would lead to a different outcome of the learning algorithm. We can make the difference between the two outcomes arbitrarily large by making some regions of the taxonomy very deep and other regions very flat. Additionally, we note that the simple graph-distance based loss works best when the taxonomy is balanced, namely, when all of the splits in the taxonomy convey roughly the same amount of information. For example, in the taxonomy of Fig. 1, the children of CLASSICAL MUSIC are VIVALDI and NON-VIVALDI, where the vast majority of classical music falls in the latter. If the correct label is NON-VIVALDI and our classifier predicts the more general label CLASSICAL MUSIC, the loss should be small, since the two labels are essentially equivalent. On the other hand, if the correct label is VIVALDI then predicting CLASSICAL MUSIC should incur a larger loss, since important detail was excluded. A simple graph-distance based loss will penalize both errors equally.

On one hand, we want to use the hierarchy to define the problem. On the other hand, we don't want arbitrary choices and unbalanced splits in the taxonomy to have a significant effect on the outcome. Can we have our cake and eat it too? Our proposed solution is to leave the taxonomy structure as-is, and to stick with a graph-distance based loss, but to introduce non-uniform *edge weights*. Namely, the loss of predicting $u$ when the true label is $y$ is defined as the sum of edge-weights along the shortest path from $u$ to $y$. We use the underlying distribution over labels to set the edge

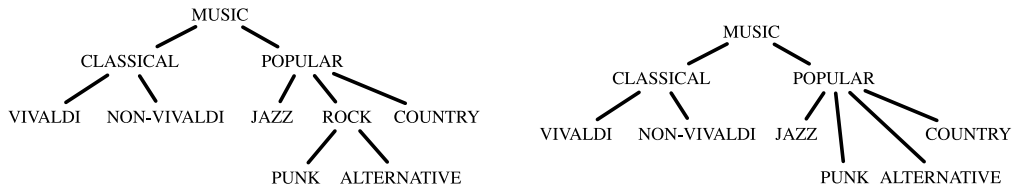

Figure 1: Two equally-reasonable label taxonomies. Note the subjective decision to include/exclude the label ROCK, and note the unbalanced split of CLASSICAL to the small class VIVALDI and the much larger class NON-VIVALDI.

weights in a way that adds balance to the taxonomy and compensates for certain arbitrary design choices. Specifically, we set edge weights using the information-theoretic notion of *conditional self-information* [7]. The weight of an edge between a label $u$ and its parent $u'$ is the log-probability of observing the label $u$ given that the example is also labeled by $u'$.

Others [19] have previously tried to use the training data to "fix" the hierarchy, as a preprocessing step to classification. However, it is unclear whether it is statistically permissible to reuse the training data twice: once to fix the hierarchy and then again in the actual learning procedure. The problem is that the preprocessing step may introduce strong statistical dependencies into our problem. These dependencies could prove detrimental to our learning algorithm, which expects to see a set of independent examples. The key to our approach is that we can estimate our distribution-dependent loss using the same data used to define it, without introducing any significant bias. It turns out that to accomplish this, we must deviate from the prevalent binomial-type estimation scheme that currently dominates machine learning and turn to a more peculiar geometric-distribution-type estimator. A binomial-type estimator essentially counts things (such as mistakes), while a geometric-type estimator measures the amount of time that passes before something occurs. Geometric-type estimators have the interesting property that they might occasionally fail, which we investigate in detail below. Moreover, we show how to control the variance of our estimate without adding bias. Since empirical estimation is the basis of supervised machine learning, we can now extrapolate hierarchical learning algorithms from our unbiased estimation technique. Specifically, we present a reduction from hierarchical classification to cost-sensitive multiclass classification, which is based on our new geometric-type estimator.

This paper is organized as follows. We formally set the problem in Sec. 2 and present our new distribution-dependent loss function in Sec. 3. In Sec. 4 we discuss how to control the variance of our empirical estimates, which is a critical step towards the learning algorithm described in Sec. 5. We conclude with a discussion in Sec. 6. We omit technical proofs due to space constraints.

## 2 Problem Setting

We now define our problem more formally. Let $\mathcal{X}$ be an instance space and let $\mathcal{T}$ be a taxonomy of labels. For simplicity, we focus on tree hierarchies. $\mathcal{T}$ is formally defined as the pair $(\mathcal{U}, \pi)$, where $\mathcal{U}$ is a finite set of labels and $\pi$ is the function that specifies the *parent* of each label in $\mathcal{U}$. $\mathcal{U}$ contains both general labels and specific labels. Specifically, we assume that $\mathcal{U}$ contains the special label ALL, and that all other labels in $\mathcal{U}$ are special cases of ALL. $\pi : \mathcal{U} \to \mathcal{U}$ is a function that defines the structure of the taxonomy by assigning a parent $\pi(u)$ to each label $u \in \mathcal{U}$. Semantically, $\pi(u)$ is a more general label than $u$ that contains $u$ as a special case. In other words, we can say that "$u$ is a specific type of $\pi(u)$". For completeness, we define $\pi(\text{ALL}) = \text{ALL}$. The *n'th generation parent function* $\pi^n : \mathcal{U} \to \mathcal{U}$ is defined by recursively applying $\pi$ to itself $n$ times. Formally

$$\pi^n(u) = \underbrace{\pi(\pi(\dots \pi(u) \dots))}_{n} .$$

For completeness, define $\pi^0$ as the identity function over $\mathcal{U}$. $\mathcal{T}$ is *acyclic*, namely, for all $u \neq \text{ALL}$ and for all $n \geq 1$ it holds that $\pi^n(u) \neq u$. The *ancestor function* $\pi^\star$, maps each label to its set of ancestors, and is defined as $\pi^\star(u) = \bigcup_{n=0}^{\infty} \{\pi^n(u)\}$. In other words, $\pi^\star(u)$ includes $u$, its parent, its parent's parent, and so on. We assume that $\mathcal{T}$ is *connected* and specifically that ALL is an ancestor

of all labels, meaning that ALL $\in \pi^\star(u)$ for all $u \in \mathcal{U}$. The inverse of the ancestor function is the *descendent function* $\tau$, which maps $u \in \mathcal{U}$ to the subset $\{u' \in \mathcal{U} : u \in \pi^\star(u')\}$. In other words, $u$ is a descendent of $u'$ if and only if $u'$ is an ancestor of $u$. Graphically, we can depict $\mathcal{T}$ as a rooted tree: $\mathcal{U}$ defines the tree nodes, ALL is the root, and $\{(u, \pi(u)) : u \in \mathcal{U} \setminus \text{ALL}\}$ is the set of edges. In this graphical representation, $\tau(u)$ includes the nodes in the subtree rooted at $u$. Using this representation, we define the *graph distance* between any two labels $d(u, u')$ as the number of edges along the path between $u$ and $u'$ in the tree. The *lowest common ancestor function* $\lambda : \mathcal{U} \times \mathcal{U} \to \mathcal{U}$ maps any pair of labels to their lowest common ancestor in the taxonomy, where "lowest" is in the sense of tree depth. Formally, $\lambda(u, u') = \pi^j(u)$ where $j = \min\{i : \pi^i(u) \in \pi^\star(u')\}$. In words, $\lambda(u, u')$ is the closest ancestor of $u$ that is also an ancestor if $u'$. It is straightforward to verify that $\lambda(u, u') = \lambda(u', u)$. The *leaves* of a taxonomy are the labels that are not parents of any other labels. We denote the set of leaves by $\mathcal{Y}$ and note that $\mathcal{Y} \subset \mathcal{U}$.

Now, let $\mathcal{D}$ be a distribution on the product space $\mathcal{X} \times \mathcal{Y}$. In other words, $\mathcal{D}$ is a joint distribution over instances and their corresponding labels. Note that we assume that the labels that occur in the distribution are always leaves of the taxonomy $\mathcal{T}$. This assumption can be made without loss of generality: if this is not the case then we can always add a leaf to each interior node, and relabel all of the examples accordingly. More formally, for each label $u \in \mathcal{U} \setminus \mathcal{Y}$, we add a new node $y$ to $\mathcal{U}$ with $\pi(y) = u$, and whenever we sample $(\mathbf{x}, u)$ from $\mathcal{D}$ then we replace it with $(\mathbf{x}, y)$. Initially, we do not know anything about $\mathcal{D}$, other than the fact that it is supported on $\mathcal{X} \times \mathcal{Y}$. We sample $m$ independent points from $\mathcal{D}$, to obtain the sample $S = \{(\mathbf{x}_i, y_i)\}_{i=1}^m$.

A *classifier* is a function $f : \mathcal{X} \to \mathcal{U}$ that assigns a label to each instance of $\mathcal{X}$. Note that a classifier is allowed to predict any label in $\mathcal{U}$, even though it knows that only leaf labels are ever observed in the real world. We feel that this property captures a fundamental characteristic of hierarchical classification: although the truth is always specific, a good hierarchical classifier will fall-back to a more general label when it cannot confidently give a specific prediction. The quality of $f$ is measured using a *loss function* $\ell : \mathcal{U} \times \mathcal{Y} \to \mathbb{R}_+$. For any instance-label pair $(\mathbf{x}, y)$, the loss $\ell(f(\mathbf{x}), y)$ should be interpreted as the penalty associated with predicting the label $f(\mathbf{x})$ when the true label is $y$. We require $\ell$ to be weakly monotonic, in the following sense: if $u'$ lies along the path from $u$ to $y$ then $\ell(u', y) \le \ell(u, y)$. Although the error indicator function, $\ell(u, y) = 1_{u \ne y}$ satisfies our requirements, it is not what we have in mind. Another fundamental characteristic of hierarchical classification problems is that not all prediction errors are equally bad, and the definition of the loss should reflect this. More specifically, if $u'$ lies along the path from $u$ to $y$ and $u$ is not semantically equivalent to $u'$, we actually expect that $\ell(u', y) < \ell(u, y)$.

## 3   A Distribution-Calibrated Loss for Hierarchical Classification

As mentioned above, we want to calibrate the hierarchical classification loss function using the distribution $\mathcal{D}$, through its empirical proxy $S$. In other words, we want $\mathcal{D}$ to differentiate between informative splits in the taxonomy and redundant ones. We follow [8] in using graph-distance to define the loss function, but instead of setting all of the edge weights to 1, we define edge weights using $\mathcal{D}$.

For each $y \in \mathcal{Y}$, let $p(y)$ be the marginal probability of the label $y$ in the distribution $\mathcal{D}$. For each $u \in \mathcal{U}$, define $p(u) = \sum_{y \in \mathcal{Y} \cap \tau(u)} p(y)$. In words, for any $u \in \mathcal{U}$, $p(u)$ is the probability of observing any descendent of $u$. We assume henceforth that $p(u) > 0$ for all $u \in \mathcal{U}$. With these definitions handy, define the weight of the edge between $u$ and $\pi(u)$ as $\log\big(p(\pi(u))/p(u)\big)$. This weight is essentially the definition of conditional self information from information theory [7].

The nice thing about this definition is that the weighted graph-distance between labels $u$ and $y$ telescopes between $u$ and $\lambda(u, y)$ and between $u$ and $\lambda(u, y)$, and becomes

$$\ell(u, y) = 2 \log\big(p(\lambda(u, y))\big) - \log\big(p(u)\big) - \log\big(p(y)\big) . \tag{1}$$

Since this loss function depends only on $u$, $y$, and $\lambda(u, y)$, and their frequencies according to $\mathcal{D}$, it is completely invariant to the the number of labels along the path from $u$ or $y$. It is also invariant to inconsistent degrees of flatness of the taxonomy in different regions. Finally, it is even invariant to the addition or subtraction of new leaves or entire subtrees, so long as the marginal distributions $p(u)$, $p(y)$, and $p(\lambda(u, y))$ remain unchanged. This loss also balances uneven splits in the taxonomy.

Recalling the example in Fig. 1 where CLASSICAL is split into VIVALDI and NON-VIVALDI, the edge to the former will have a very high weight, whereas the edge to the latter will have a weight close to zero.

Now, define the risk of a classifier $h$ as $\mathcal{R}(f) = \mathbb{E}_{(X,Y)\sim\mathcal{D}}[\ell(f(X), Y)]$, the expected loss over examples sampled from $\mathcal{D}$. Our goal is to obtain a classifier with a small risk. However, before we tackle the problem of finding a low risk classifier, we address the intermediate task of estimating the risk of a given classifier $f$ using the sample $S$. The solution is not straightforward since we cannot even compute the loss on an individual example, $\ell(f(\mathbf{x}_i), y_i)$, as this requires knowledge of $\mathcal{D}$. A naive way to estimate $\ell(f(\mathbf{x}_i), y_i)$ using the sample $S$ is to first estimate each $p(y)$ by $\sum_{i=1}^{m} 1_{y_i=y}$, and to plug these values into the definition of $\ell$. This estimator tends to suffer from a strong bias, due to the non-linearity of the logarithm, and is considered to be unreliable[1]. Instead, we want an unbiased estimator.

First, we write the definition of risk more explicitly using the definition of the loss function in Eq. (1). Define $q(f, u) = \Pr(f(X) = u)$, the probability that $f$ outputs $u$ when $X$ is drawn according to the marginal distribution of $\mathcal{D}$ over $\mathcal{X}$. Also define $r(f, u) = \Pr(\lambda(f(X), Y) = u)$, the probability that the lowest common ancestor of $f(X)$ and $Y$ is $u$, when $(X, Y)$ is drawn from $\mathcal{D}$. $\mathcal{R}(f)$ can be rewritten as

$$\mathcal{R}(f) = \sum_{u \in \mathcal{U}} \big(2r(f, u) - q(f, u)\big) \log(p(u)) - \sum_{y \in \mathcal{Y}} p(y) \log\big(p(y)\big) \ . \tag{2}$$

Notice that the second term in the definition of risk is a constant, independent of $f$. This constant is simply $H(Y)$, the Shannon entropy [7] of the label distribution. Our ultimate goal is to compare the risk values of different classifiers and to choose the best one, so we don't really care about this constant, and we can discard it henceforth. From here on, we focus on estimating the augmented risk $\bar{\mathcal{R}}(f) = \mathcal{R}(f) - H(Y)$.

The main building block of our estimator is the estimation technique presented in [14]. Assume for a moment that the sample $S$ is infinite. Recall that the harmonic number $h_n$ is defined as $\sum_{i=1}^{n} \frac{1}{i}$, with $h_0 = 0$. Define the random variables $A_i$ and $B_i$ as follows

$$A_i = \min\{j \in \mathbb{N} \ : \ y_{i+j} \in \tau(f(\mathbf{x}_i))\} - 1$$
$$B_i = \min\big\{j \in \mathbb{N} \ : \ y_{i+j} \in \tau\big(\lambda(f(\mathbf{x}_i), y_i)\big)\big\} - 1$$

For example, $A_1 + 2$ is the index of the first example after $(\mathbf{x}_1, y_1)$ whose label is contained in the subtree rooted at $f(\mathbf{x}_1)$, and $B_1 + 2$ is the index of the first example after $(\mathbf{x}_1, y_1)$ whose label is contained in the subtree rooted at $\lambda(f(\mathbf{x}_1), y_1)$. Note that $B_i \leq A_i$, since $\lambda(u, y)$ is, by definition, an ancestor of $u$, so $y' \in \tau(u)$ implies $y' \in \tau(\lambda(u, y))$. Next, define the random variable $L_1 = h_{A_1} - 2h_{B_1}$.

**Theorem 1.** $L_1$ is an unbiased estimator of $\bar{\mathcal{R}}(f)$.

*Proof.* We have that

$$\mathbb{E}\big[L_1 \, \big| \, f(X_1) = u, Y_1 = y\big] = p(u) \sum_{j=0}^{\infty} h_j \big(1 - p(u)\big)^j - 2p\big(\lambda(u, y)\big) \sum_{j=0}^{\infty} h_j \big(1 - p(\lambda(u, y))\big)^j \ .$$

Using the fact that for any $\alpha \in [0, 1)$ it holds that $\sum_{n=0}^{\infty} h_n \alpha^n = -\frac{\log(1-\alpha)}{1-\alpha}$ we get, $\mathbb{E}[L_1 | f(X_1) = u, Y_1 = y] = -\log\big(p(u)\big) + 2\log\big(p(\lambda(u, y))\big)$. Therefore,

$$\begin{aligned}
\mathbb{E}[L_1] &= \sum_{u \in \mathcal{U}} \sum_{y \in \mathcal{Y}} \Pr(f(X) = u, Y = y) \, \mathbb{E}[L_1 | f(X_1) = u, Y_1 = y] \\
&= \sum_{u \in \mathcal{U}} \big(2r(f, u) - q(f, u)\big) \log\big(p(u)\big) = \bar{\mathcal{R}}(f) \ .
\end{aligned}$$ $\square$

We now recall that our sample $S$ is actually of finite size $m$. The problem that now occurs is that $A_1$ and $B_1$ are not well defined when $f(X_1)$ does not appear anywhere in $Y_2, \ldots, Y_m$. When this happens, we say that the estimator $L_1$ *fails*. If $f$ outputs a label $u$ with $p(u) = 0$ then $L_1$ will fail

with probability 1. On the other hand, the probability of failure is negligible when $m$ is large enough, and when $f$ does not output labels with tiny probabilities. Formally, let $\beta(f) = \min_{u:q(f,u)>0} p(u)$ be the smallest probability of any label that $f$ outputs.

**Theorem 2.** *The probability of failure is at most $e^{-(m-1)\beta(f)}$.*

The estimator $\mathbb{E}[L_1|no\text{-}fail]$ is no longer an unbiased estimator of $\bar{\mathcal{R}}(f)$, but the bias is small. Specifically, since we are after a classifier $f$ with a small risk, we prove an upper-bound on $\bar{\mathcal{R}}(f)$.

**Theorem 3.** *It holds that $\mathbb{E}[L_1|no\text{-}fail] \geq \bar{\mathcal{R}}(f) - \frac{(m-1)e^{-\beta(f)(m-1)}}{\beta^2(f)}$.*

For example, with $\beta = 0.01$ and $m = 2500$, the bias term in Thm. 3 is less than 0.0004. With $m = 5000$ it is already less than $10^{-14}$.

## 4 Decreasing the Variance of the Estimator

Say that we have $k$ classifiers and we want to choose the best one. The estimator $L_1$ suffers from an unnecessarily high variance because it typically uses a short prefix of the sample $S$ and wastes the remaining examples. To reliably compare $k$ empirical risk estimates, we need to reduce the variance of each estimator. The exact value of $\mathrm{Var}(L_1)$ depends on the distributions $p$, $q$, and $r$ in a non-trivial way, but we can give a simple upper-bound on $\mathrm{Var}(L_1)$ in terms of $\beta(f)$.

**Theorem 4.** $\mathrm{Var}(L_1) \leq -9\log\left(\beta(f)\right) + 9\log^2\left(\beta(f)\right)$.

We reduce the variance of the estimator by repeating the estimation multiple times, without reusing any sample points. Formally, define $S_1 = 1$, and define for all $i \geq 2$ the random variables $S_i = S_{i-1} + A_{S_{i-1}} + 2$, and $L_i = h_{A_{S_i}} - 2h_{B_{S_i}}$. In words: the first estimator $L_1$ starts at $S_1 = 1$ and uses $A_1 + 2$ examples, namely, the examples $1, \ldots, (A_1 + 2)$. Now, $S_2 = A_1 + 3$ is the first untouched example in the sequence. The second estimator, $L_2$ starts at example $S_2$ and uses $A_{S_2} + 2$ examples, namely, the examples $S_2, \ldots, (S_2 + A_{S_2} + 1)$, and so on. If we had an infinite sample and chose some threshold $t$, the random variables $L_1, \ldots, L_t$ would all be unbiased estimators of $\bar{\mathcal{R}}(f)$, and therefore the aggregate estimator $L = \frac{1}{t}\sum_{i=1}^{t} L_i$ would also be an unbiased estimate of $\bar{\mathcal{R}}(f)$. Since $L_1, \ldots, L_t$ are also independent, the variance of the aggregate estimator would be $\frac{1}{t}\mathrm{Var}(L_1)$.

In the finite-sample case, aggregating multiple estimators is not as straightforward. Again, the event where the estimation fails introduces a small bias. Additionally, the number of independent estimations that fit in a sample of fixed size $m$ is itself a random variable $T$. Moreover, the value of $T$ depends on the value of the risk estimators. In other words, if $L_1, L_2, \ldots$ take large values then $T$ will take a small value. The precise definition of $T$ should be handled with care, to ensure that the individual estimators remain independent and that the aggregate estimator maintains a small bias. For example, the first thing that comes to mind is to set $T$ to be the largest number $t$ such that $S_t \leq m$ - this is a bad idea. To see why, note that if $T = 2$ and $A_1 = m - 4$ then we know with certainty that $A_{S_2} = 0$. This clearly demonstrates a strong statistical dependence between $L_1, L_2$ and $T$, which both interferes with the variance reduction and introduces a bias. Instead, we define $T$ as follows: choose a positive integer $l \leq m$ and set $T$ using the last $l$ examples in $S$, as follows, set

$$T = \min\{t \in \mathbb{N} : S_{t+1} \geq m - l\}. \tag{3}$$

In words, we think of the last $l$ examples in $S$ as the "landing strip" of our procedure: we keep jumping forward in the sequence of samples, from $S_1$ to $S_2$, to $S_3$, and so on, until the first time we land on the landing strip. Our new failure scenario occurs when our last jump overshoots the strip, and no $S_i$ falls on any one of the last $l$ examples. If $L$ does not fail, define the aggregate estimator as $L = \sum_{i=1}^{T} L_i$. Note that we are summing $L_i$ rather than averaging them; we explain this later on.

**Theorem 5.** *The probability of failure of the estimator $L$ is at most $e^{-l\beta(f)}$.*

We now prove that our definition of $T$ indeed decreases the variance without adding bias. We give a simplified version of the analysis, assuming that $S$ is infinite, and assuming that the limit $m$ is merely a recommendation. In other words, $T$ is still defined as before, but estimation never fails, even in the rare case where $S_T + A_{S_T} + 1 > m$ (the index of the last example used in the estimation exceeds the predefined limit $m$). We note that a very similar theorem can be stated in the finite-sample case,

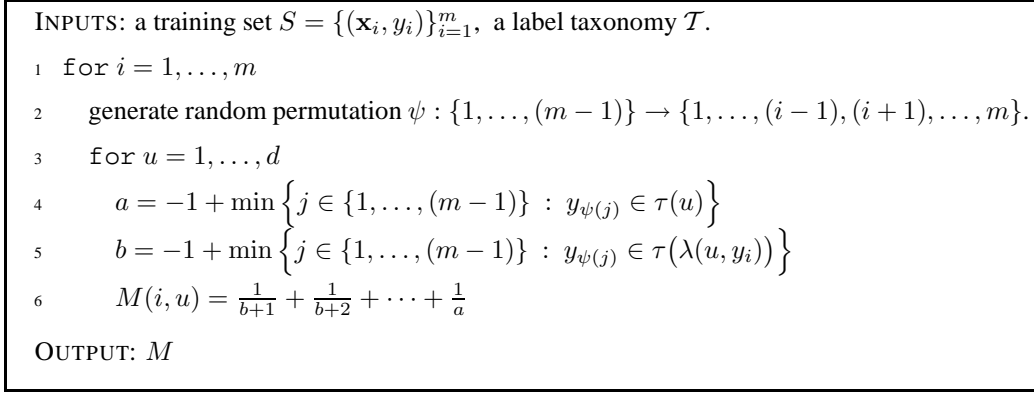

INPUTS: a training set $S = \{(\mathbf{x}_i, y_i)\}_{i=1}^m$, a label taxonomy $\mathcal{T}$.

1  for $i = 1, \ldots, m$

2    generate random permutation $\psi : \{1, \ldots, (m-1)\} \to \{1, \ldots, (i-1), (i+1), \ldots, m\}$.

3    for $u = 1, \ldots, d$

4      $a = -1 + \min \left\{ j \in \{1, \ldots, (m-1)\} \ : \ y_{\psi(j)} \in \tau(u) \right\}$

5      $b = -1 + \min \left\{ j \in \{1, \ldots, (m-1)\} \ : \ y_{\psi(j)} \in \tau\big(\lambda(u, y_i)\big) \right\}$

6      $M(i, u) = \frac{1}{b+1} + \frac{1}{b+2} + \cdots + \frac{1}{a}$

OUTPUT: $M$

Figure 2: A reduction from hierarchical multiclass to cost-sensitive multiclass.

at the price of a significantly more complicated analysis. The complication stems from the fact that we are estimating the risk of $k$ classifiers simultaneously, and the failure of one estimator depends on the values of the other estimators. We allow ourselves to ignore failures because they occur with such small probability, and because they introduce an insignificant bias.

**Theorem 6.** *Assuming that $S$ is infinite, but $T$ is still defined as in Eq. (3), it holds that $\mathbb{E}\big[L\big] = \mathbb{E}\big[T\big]\bar{\mathcal{R}}(f)$ and $\mathrm{Var}(L) \leq \mathbb{E}[T]\sigma^2$, where $\sigma^2 = \mathrm{Var}\big(L_i\big)$.*

The proof follows from variations on Wald's theorem [15].

Recall that we have $k$ competing classifiers, $f_1, \ldots, f_k$, and we want to choose one with a small risk. We overload our notation to support multiple concurrent estimations, and define $T(f_j)$ as the stopping time (previously defined as $T$ in Eq. (3)) of the estimation process for $\bar{\mathcal{R}}(f_j)$. Also let $L_i(f_j)$ be the $i$'th unbiased estimator of $\bar{\mathcal{R}}(f_j)$. To conduct a fair comparison of the $k$ classifiers, we redefine $T = \min_{j=1,\ldots,k} T(f_j)$, and let $L(f_j) = \sum_{i=1}^T L_i(f_j)$. In other words, we aggregate the same number of estimators for each classifier. We then choose the classifier with the smallest risk estimate, $\arg \min L(F_j)$. Theorem 6 still holds for each individual classifier because the new definition of $T$ remains a stopping time for each of the individual estimation processes. Although we may not know the exact value of $\mathbb{E}[T]$, it is just a number that we can use to reason about the bias and the variance of $L$. We note that finding $j$ that minimizes $L(f_j)$ is equivalent to finding $j$ that minimizes $L(f_j)/\mathbb{E}[T]$. The latter, according to Thm. 6, is an unbiased estimate of $\bar{\mathcal{R}}(f)$. Moreover, the variance of each $L(f_j)/\mathbb{E}[T]$ is $\mathrm{Var}\left(L(f_j)/E[T]\right) = \sigma^2/\mathbb{E}[T]$, so the effective variance of our unbiased estimate decreases like $1/\mathbb{E}[T]$, which is what we would expect. Using the one-tailed Chebyshev inequality [11], we get that for any $\epsilon > 0$, $\mathrm{Pr}\left(\bar{\mathcal{R}}(f_j) \geq L(f_j) + \epsilon\right) < \sigma^2/(\sigma^2 + \mathbb{E}[T]\epsilon^2)$. The bound holds uniformly for all $k$ classifiers with probability $k\sigma^2/(\sigma^2 + \mathbb{E}[T]\epsilon^2)$ (using the union bound). The variance of the estimation depends on $\mathbb{E}[T]$, and we expect $\mathbb{E}[T]$ to grow linearly with $m$. For example we can prove the following crude lower-bound.

**Theorem 7.** $\mathbb{E}[T] \geq (m - l)/c$, *where* $c = k + \sum_{j=1}^k 1/\beta(f_j)$.

## 5 Reducing Hierarchical Classification to Cost-Sensitive Classification

In this section, we propose a method for learning low-risk hierarchical classifiers, using our new definition of risk. More precisely, we describe a reduction from hierarchical classification to *cost-sensitive multiclass classification*. The appeal of this approach is the abundance of existing cost-sensitive learning algorithms. This reduction is itself an algorithm whose input is a training set of $m$ examples and a taxonomy over $d$ labels, and whose output is a $d \times m$ matrix of non-negative reals, denoted by $M$. Entry $M(i, j)$ is the cost of classifying example $i$ with label $j$. This cost matrix, and the original training set, are given to a cost-aware multiclass learning algorithm, which attempts to find a classifier $f$ with a small empirical loss $\sum_{i=1}^m M(i, f(\mathbf{x}_i))$.

For example, a common approach to multiclass problems is to train a model $f_u : \mathcal{X} \to \mathbb{R}$ for each label $u \in \mathcal{U}$ and to define the classifier $f(\mathbf{x}) = \arg\max_{u \in \mathcal{U}} f_u(\mathbf{x})$. An SVM-flavored way to train a cost sensitive classifier is to assume that the functions $f_u$ live in a Hilbert space, and to minimize

$$\sum_{u=1}^{d} \|f_u\|^2 + C \sum_{i=1}^{m} \sum_{u \neq y_i} \left[ M(i,u) + f_u(\mathbf{x}_i) - f_{y_i}(\mathbf{x}_i) \right]_+ , \qquad (4)$$

where $C > 0$ is a parameter and $[\alpha]_+ = \max\{0, \alpha\}$. The first term is a regularizer and the second is an empirical loss, justified by the fact that $M(i, f(\mathbf{x}_i)) \leq \sum_{u \neq y_i} \left[ M(i,u) + f_u(\mathbf{x}_i) - f_{y_i}(\mathbf{x}_i) \right]_+$.

Coming back to the reduction algorithm, we generate $M$ using the procedure outlined in Fig. 2. Based on the analysis of the previous sections, it is easy to see that, for all $i$, $M(i, f(\mathbf{x}_i))$ is an unbiased estimator of the risk $\bar{\mathcal{R}}(f)$. This holds even if $\psi$ (as defined in Fig. 2) is a fixed function, because the training set is assumed to be i.i.d. Therefore, $\frac{1}{m} \sum M(i, f(\mathbf{x}_i))$ is also an unbiased estimator of $\bar{\mathcal{R}}(f)$. The cost-sensitive learning algorithm will try to minimize this empirical estimate. The purpose of the random permutation at each step is to hopefully decrease the variance of the overall estimate, by decreasing the dependencies between the different individual estimators. We profess that a rigorous analysis of the variance of this estimator is missing from this work. Ideally, we would like to show that, with high probability, the empirical estimate $\frac{1}{m} \sum M(i, f(\mathbf{x}_i))$ is $\epsilon$-close to its expectation of $\bar{\mathcal{R}}(f)$, uniformly for all classifiers $f$ in our function class. This is a challenging problem due to the complex dependencies in the estimator.

The learning algorithm used to solve this problem can (and should) use the hierarchical structure to guide its search for a good classifier. Our reduction to an unstructured cost-sensitive problem should not be misinterpreted as a recommendation not to use the structure in the learning process. For example, following [10, 8], we could augment the SVM approach described in Eq. (4) by replacing the unstructured regularizer $\sum_{u=1}^{d} \|f_u\|^2$ with the structured regularizer $\sum_{u=1}^{d} \|f_u - f_{\pi(u)}\|^2$, where $\pi(u)$ is the parent label of $u$. [8] showed significant gains on hierarchical problems using this regularizer.

## 6 Discussion

We started by taking a step back from the typical setup of a hierarchical classification machine learning problem. As a consequence, our focus was on the fundamental aspects of the hierarchical problem definition, rather than on the equally important algorithmic issues. Our discussion was restricted to the simplistic model of single-label hierarchical classification with single-linked taxonomies, and our first goal going forward is to relax these assumptions.

We point out that many of the theorems proven in this paper depend on the value of $\beta(f)$, which is defined as $\min_{u:q(u)>0} p(u)$. Specifically, if $f$ occasionally outputs a very rare label, then $\beta(f)$ is tiny and much of our analysis breaks down. This provides a strong indication that an empirical estimate of $\beta(f)$ would make a good regularization term in a hierarchical learning scheme. In other words, we should deter the learning algorithm from choosing a classifier that predicts very rare labels. As mentioned in the introduction, the label taxonomy provides the perfect mechanism for backing off and predicting a more common and less risky ancestor of that label.

We believe that our work is significant in the broader context of *structured learning*. Most structured learning algorithms blindly trust the structure that they are given, and arbitrary design choices are likely to appear in many types of structured learning. The idea of using the data distribution to calibrate, correct, and balance the side-information extends to other structured learning scenarios. The geometric-type estimation procedure outlined in this paper may play an important role in those settings as well.

### Acknowledgment

The author would like to thank Paul Bennett for his suggestion of the loss function for its information theoretic properties, reduction to a tree-weighted distance, and ability to capture other desirable characteristics of hierarchical loss functions like weak monotonicity. The author also thanks Ohad Shamir, Chris Burges, and Yael Dekel for helpful discussions.

## Footnotes

[1]The interested reader is referred to the extensive literature on the closely related problem of estimating the entropy of a distribution from a finite sample.

# References

[1] The Library of Congress Classification. http://www.loc.gov/aba/cataloging/classification/.

[2] The Open Directory Project. http://www.dmoz.org/about.html.

[3] L. Cai and T. Hofmann. Hierarchical document categorization with support vector machines. In *13th ACM Conference on Information and Knowledge Management*, 2004.

[4] N. Cesa-Bianchi, C. Gentile, and L. Zaniboni. Hierarchical classification: combining bayes with svm. In *Proceedings of the 23rd International Conference on Machine Learning*, 2006.

[5] N. Cesa-Bianchi, C. Gentile, and L. Zaniboni. Incremental algorithms for hierarchical classification. *Journal of Machine Learning Research*, 7:31–54, 2007.

[6] The Gene Ontology Consortium. Gene ontology: tool for the unification of biology. *Nature Genetics*, 25:25–29, 2000.

[7] T. M. Cover and J. A. Thomas. *Elements of Information Theory*. Wiley, 1991.

[8] O. Dekel, J. Keshet, and Y. Singer. Large margin hierarchical classification. In *Proceedings of the Twenty-First International Conference on Machine Learning*, 2004.

[9] S. T. Dumais and H. Chen. Hierarchical classification of Web content. In *Proceedings of SIGIR-00*, pages 256–263, 2000.

[10] T. Evgeniou, C.Micchelli, and M. Pontil. Learning multiple tasks with kernel methods. *Journal of Machine Learning Research*, 6:615–637, 2005.

[11] W. Feller. *An Introduction to Probability and its Applications*, volume 2. John Wiley and Sons, second edition, 1970.

[12] D. Koller and M. Sahami. Hierarchically classifying docuemnts using very few words. In *Machine Learning: Proceedings of the Fourteenth International Conference*, pages 171–178, 1997.

[13] A. K. McCallum, R. Rosenfeld, T. M. Mitchell, and A. Y. Ng. Improving text classification by shrinkage in a hierarchy of classes. In *Proceedings of ICML-98*, pages 359–367, 1998.

[14] S. Montgomery-Smith and T. Schurmann. Unbiased estimators for entropy and class number.

[15] S.M. Ross and E.A. Pekoz. *A second course in probability theory*. 2007.

[16] E. Ruiz and P. Srinivasan. Hierarchical text categorization using neural networks. *Information Retrieval*, 5(1):87–118, 2002.

[17] C. Shirky. Ontology is overrated: Categories, links, and tags. In *O'Reilly Media Emerging Technology Conference*, 2005.

[18] A. S. Weigend, E. D. Wiener, and J. O. Pedersen. Exploiting hierarchy in text categorization. *Information Retrieval*, 1(3):193–216, 1999.

[19] J. Zhang, L. Tang, and H. Liu. Automatically adjusting content taxonomies for hierarchical classification. In *Proceedings of the Fourth Workshop on Text Mining, SDM06*, 2006.

